# Segmentation Circuits Using Constrained Optimization

John G. Harris*
MIT AI Lab
545 Technology Sq., Rm 767
Cambridge, MA 02139

## Abstract

A novel segmentation algorithm has been developed utilizing an absolute-value smoothness penalty instead of the more common quadratic regularizer. This functional imposes a piece-wise constant constraint on the segmented data. Since the minimized energy is guaranteed to be convex, there are no problems with local minima and no complex continuation methods are necessary to find the unique global minimum. By interpreting the minimized energy as the generalized power of a nonlinear resistive network, a continuous-time analog segmentation circuit was constructed.

## 1   INTRODUCTION

Analog hardware has obvious advantages in terms of its size, speed, cost, and power consumption. Analog chip designers, however, should not feel constrained to mapping existing digital algorithms to silicon. Many times, new algorithms must be adapted or invented to ensure efficient implementation in analog hardware. Novel analog algorithms embedded in the hardware must be simple and obey the natural constraints of physics. Much algorithm intuition can be gained from experimenting with these continuous-time nonlinear systems. For example, the algorithm described in this paper arose from experimentation with existing analog segmentation hardware. Surprisingly, many of these "analog" algorithms may prove useful even if a computer vision researcher is limited to simulating the analog hardware on a digital computer [7].

## 2   ABSOLUTE-VALUE SMOOTHNESS TERM

Rather than deal with systems that have many possible stable states, a network that has a unique stable state will be studied. Consider a network that minimizes:

$$E(u) = \frac{1}{2} \sum_i (d_i - u_i)^2 + \lambda \sum_i |u_{i+1} - u_i| \qquad (2)$$

The absolute-value function is used for the smoothness penalty instead of the more familiar quadratic term. There are two intuitive reasons why the absolute-value penalty is an improvement over the quadratic penalty for piece-wise constant segmentation. First, for large values of $|u_i - u_{i+1}|$, the penalty is not as severe, which means that edges will be smoothed less. Second, small values of $|u_i - u_{i+1}|$ are penalized more than they are in the quadratic case, resulting in a flatter surface between edges. Since no complex continuation or annealing methods are necessary to avoid local minima, this computational model is of interest to vision researchers independent of any hardware implications.

This method is very similar to constrained optimization methods discussed by Platt [14] and Gill [4]. Under this interpretation, the problem is to minimize $\sum (d_i - u_i)^2$ with the constraint that $u_i = u_{i+1}$ for all $i$. Equation 1 is an instance of the penalty method, as $\lambda \to \infty$, the constraint $u_i = u_{i+1}$ is fulfilled exactly. The absolute-value value penalty function given in Equation 2 is an example of a nondifferential penalty. The constraint $u_i = u_{i+1}$ is fulfilled exactly for a finite value of $\lambda$. However, unlike typical constrained optimization methods, this application requires some of these "exact" constraints to fail (at discontinuities) and others to be fulfilled.

This algorithm also resembles techniques in robust statistics, a field pioneered and formalized by Huber [9]. The need for robust estimation techniques in visual processing is clear since, a single outlier may cause wild variations in standard regularization networks which rely on quadratic data constraints [17]. Rather than use the quadratic data constraints, robust regression techniques tend to limit the influence of outlier data points.[2] The absolute-value function is one method commonly used to reduce outlier susceptability. In fact, the absolute-value network developed in this paper is a robust method if discontinuities in the data are interpreted as outliers. The line process or resistive fuse networks can also be interpreted as robust methods using a more complex influence functions.

## 3   ANALOG MODELS

As pointed out by Poggio and Koch [15], the notion of minimizing power in linear networks implementing quadratic "regularized" algorithms must be replaced by the more general notion of minimizing the total resistor co-content [13] for nonlinear networks. For a voltage-controlled resistor characterized by $I = f(V)$, the co-content is defined as

$$J(V) = \int_0^V f(V')dV' \qquad (3)$$

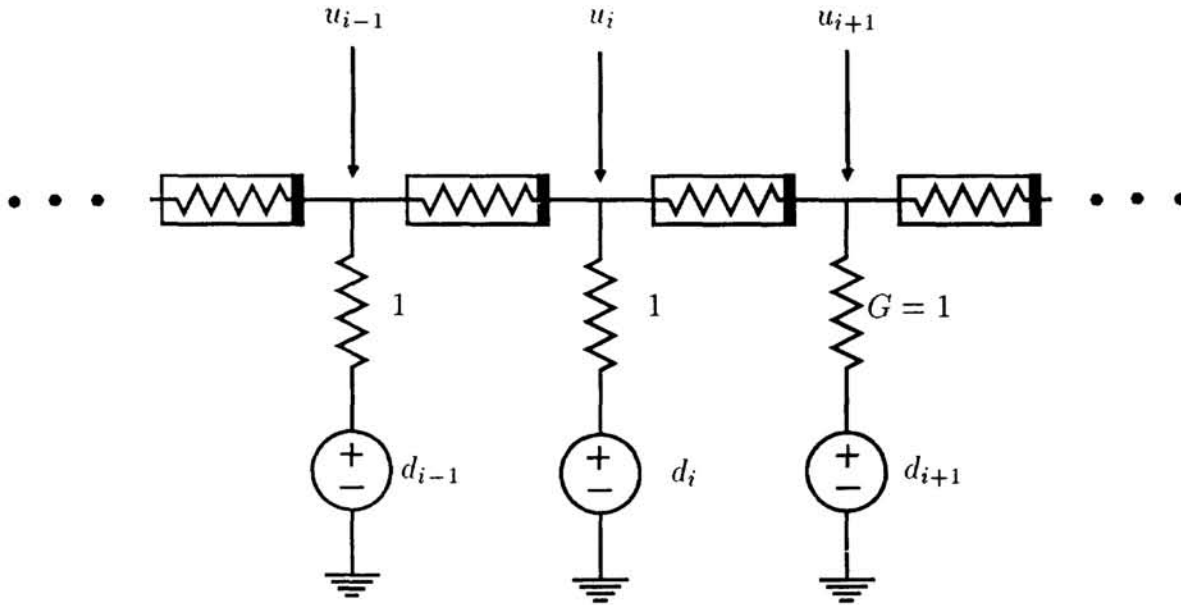

Figure 1: Nonlinear resistive network for piece-wise constant segmentation.

One-dimensional surface interpolation from dense data will be used as the model problem in this paper, but these techniques generalize to sparse data in multiple dimensions. A standard technique for smoothing or interpolating noisy inputs $d_i$ is to minimize an energy[1] of the form:

$$E(u) = \sum_i (d_i - u_i)^2 + \lambda \sum_i (u_{i+1} - u_i)^2 \tag{1}$$

The first term ensures that the solution $u_i$ will be close to the data while the second term implements a smoothness constraint. The parameter $\lambda$ controls the tradeoff between the degree of smoothness and the fidelity to the data. Equation 1 can be interpreted as a regularization method [1] or as the power dissipated the linear version of the resistive network shown in Figure 1 [16].

Since the energy given by Equation 1 oversmoothes discontinuities, numerous researchers (starting with Geman and Geman [3]) have modified Equation 1 with line processes and successfully demonstrated piece-wise smooth segmentation. In these methods, the resultant energy is nonconvex and complex annealing or continuation methods are required to converge to a good local minima of the energy space. This problem is solved using probabilistic [11] or deterministic annealing techniques [2, 10]. Line-process discontinuities have been successfully demonstrated in analog hardware using resistive fuse networks [5], but continuation methods are still required to find a good solution [6].

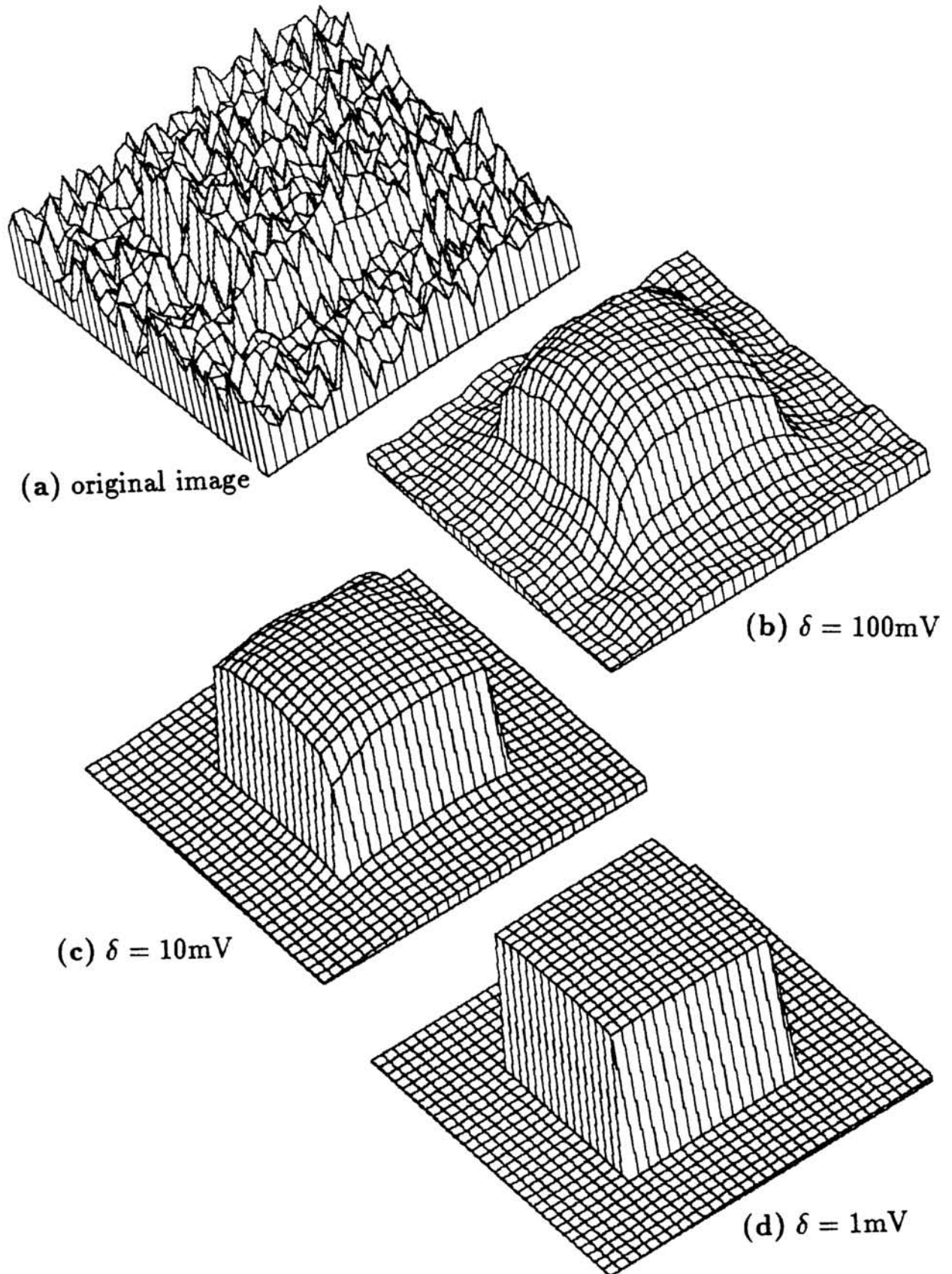

(a) original image

(b) $\delta = 100\mathrm{mV}$

(c) $\delta = 10\mathrm{mV}$

(d) $\delta = 1\mathrm{mV}$

Figure 2: Various examples of tiny-tanh network simulation for varying $\delta$. The I-V characteristic of the saturating resistors is $I = \lambda \tanh(V/\delta)$. (a) shows a synthetic 1.0V tower image with additive Gaussian noise of $\sigma = 0.3$V which is input to the network. The network outputs are shown in Figures (b) $\delta = 100$mV, (c) $\delta = 10$mV and (d) $\delta = 1$mV. For all simulations $\lambda = 1$.

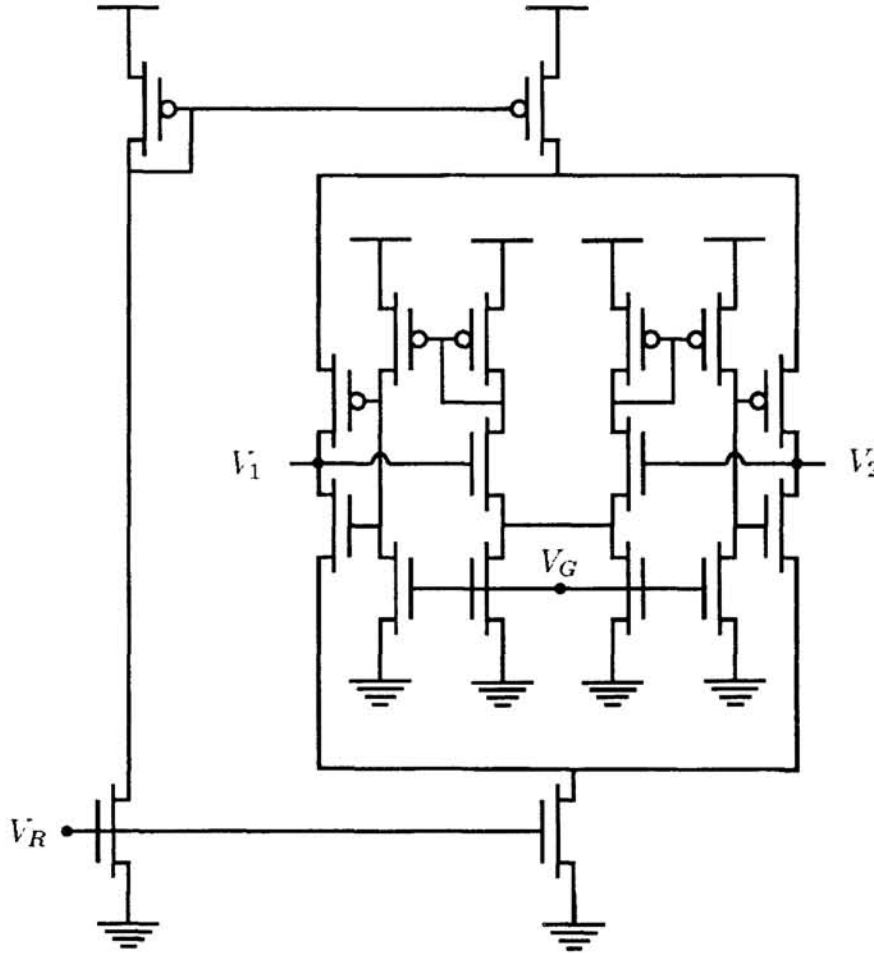

Figure 3: Tiny tanh circuit. The saturating tanh characteristic is measured between nodes $V_1$ and $V_2$. Controls $V_R$ and $V_G$ set the conductance and saturation voltage for the device.

For a linear resistor, $I = GV$, the co-content is given by $\frac{1}{2}GV^2$, which is half the dissipated power $P = GV^2$.

The absolute-value functional in Equation 2 is not strictly convex. Also, since the absolute-value function is nondifferentiable at the origin, hardware and software methods of solution will be plagued with instabilities and oscillations. We approximate Equation 2 with the following well-behaved convex co-content:

$$\frac{1}{2}\sum_i (u_i - d_i)^2 + \sum_i \int_0^{u_{i+1}-u_i} \lambda \tanh(v/\delta)dv \qquad (4)$$

The co-content becomes the absolute-value cost function in Equation 2 in the limiting case as $\delta \to 0$. The derivative of Equation 2 yields Kirchoff's current equation at each node of the resistive network in Figure 1:

$$(u_i - d_i) + \lambda \tanh(\frac{u_i - u_{i+1}}{\delta}) + \lambda \tanh(\frac{u_i - u_{i-1}}{\delta}) = 0 \qquad (5)$$

Therefore, construction of this network requires a nonlinear resistor with a hyperbolic tangent I-V characteristic with an extremely narrow linear region. For this

reason, this element is called the *tiny-tanh* resistor. This saturating resistor is used as the nonlinear element in the resistive network shown in Figure 1. Its I-V characteristic is $I = \lambda \tanh(V/\delta)$. It is well-known that any circuit made of independent voltage sources and two-terminal resistors with strictly increasing I-V characteristics has a unique stable state.

## 4    COMPUTER SIMULATIONS

Figure 2a shows a synthetic 1.0V tower image with additive Gaussian noise of $\sigma = 0.3$V. Figure 2b shows the simulated result for $\delta = 100$mV and $\lambda = 1$. As Mead has observed, a network of saturating resistors has a limited segmentation effect [12]. Unfortunately, as seen in the figure, noise is still evident in the output, and the curves on either side of the step have started to slope toward one another. As $\lambda$ is increased to further smooth the noise, the two sides of the step will blend together into one homogeneous region. However, as the width of the linear region of the saturating resistor is reduced, network segmentation properties are greatly enhanced. Segmentation performance improves for $\delta = 10$mV shown in Figure 2c and further improves for $\delta = 1mV$ in Figure 2d. The best segmentation occurs when the I-V curve resembles a step function, and co-content, therefore, approximates an absolute-value. Decreasing $\delta$ less than 1mV shows no discernible change in the output.[3]

One drawback of this network is that it does not recover the exact heights of input steps. Rather it subtracts a constant from the height of each input. It is straightforward to show that the amount each uniform region is pulled towards the background is given by $\lambda$(perimeter/area) [7]. Significant features with large area/perimeter ratios will retain their original height. Noise points have small area/perimeter ratios and therefore will be pulled towards the background. Typically, the exact values of the heights are less important than the location of the discontinuities. Furthermore, it would not be difficult to construct a two-stage network to recover the exact values of the step heights if desired. In this scheme a tiny-tanh network would control the switches on a second fuse network.

## 5    ANALOG IMPLEMENTATION

Mead has constructed a CMOS saturating resistor with an I-V characteristic of the form $I = \lambda \tanh(V/\delta)$, where delta must be larger than 50mV because of fundamental physical limitations [12]. Simulation results from section 4 suggest that for a tower of height $h$ to be segmented, $h/\delta$ must be at least on the order of 1000. Therefore a network using Mead's saturating resistor ($\delta = 50$mV) could segment a tower on the order of 50V, which is much too large a voltage to input to these chips. Furthermore, since we are typically interested in segmenting images into more than two levels even higher voltages would be required. The tiny-tanh circuit (shown in Figure 3) builds upon an older version of Mead's saturating resistor [18] using a gain stage to decrease the linear region of the device. This device can be made to saturate at voltages as low as 5mV.

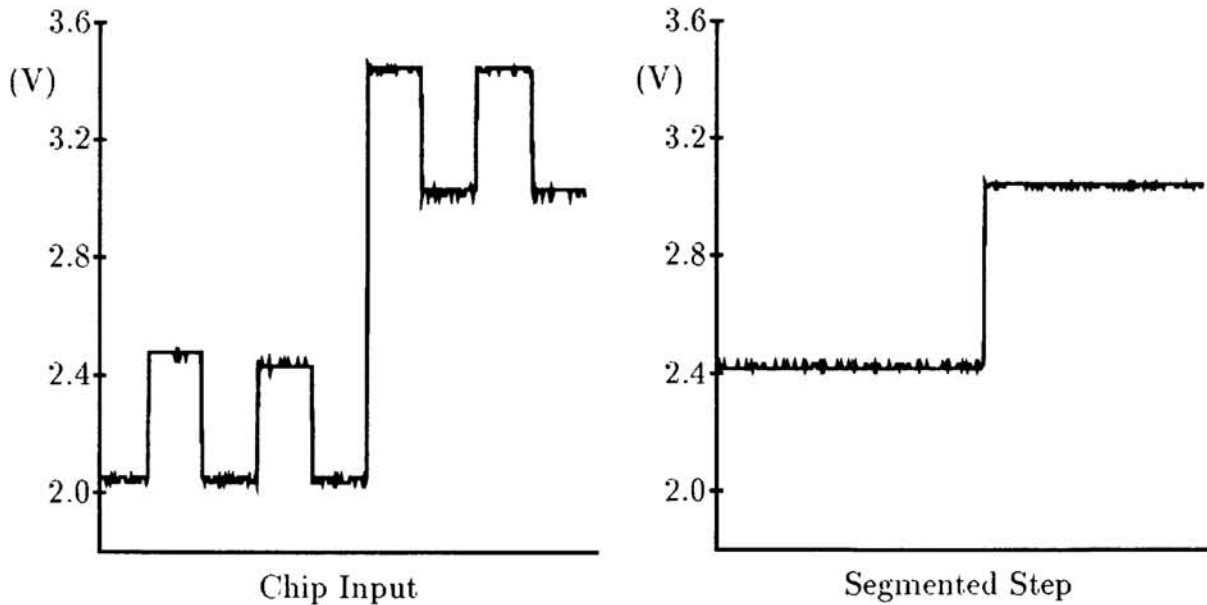

Figure 4: Measured segmentation performance of the tiny-tanh network for a step. The input shown on the left is about a 1V step. The output shown on the right is a segmented step about 0.5V in height.

By implementing the nonlinear resistors in Figure 1 with the tiny-tanh circuit, a 1D segmentation network was successfully fabricated and tested. Figure 4 shows the segmentation which resulted when a step (about 1V) was scanned into the chip. The segmented step has been reduced to about 0.5V. No special annealing methods were necessary because a convex energy is being minimized.

# 6   CONCLUSION

A novel energy functional was developed for piece-wise constant segmentation.[4] This computational model is of interest to vision researchers independent of any hardware implications, because a convex energy is minimized. In sharp contrast to previous solutions of this problem, no complex continuation or annealing methods are necessary to avoid local minima. By interpreting this Lyapunov energy as the co-content of a nonlinear circuit, we have built and demonstrated the tiny-tanh network, a continuous-time segmentation network in analog VLSI.

**Acknowledgements**

Much of this work was perform at Caltech with the support of Christof Koch and Carver Mead. A Hughes Aircraft graduate student fellowship and an NSF postdoctoral fellowship are gratefully acknowledged.

## Footnotes

*A portion of this work is part of a Ph.D dissertation at Caltech [7].

[2]Outlier detection techniques have been mapped to analog hardware [8].

[1]The term *energy* is used throughout this paper as a cost functional to be minimized. It does not necessarily relate to any true energy dissipated in the real world.

[3]These simulations were also used to smooth and segment noisy depth data from a correlation-based stereo algorithm run on real images [7].

[4]This work has also been extended to segment piece-wise linear regions, instead of the purely piece-wise constant processing discussed in this paper [7].

# References

[1] M. Bertero, T. Poggio, and V. Torre. Ill-posed problems in early vision. *Proc. IEEE*, 76:869–889, 1988.

[2] A. Blake and A. Zisserman. *Visual Reconstruction*. MIT Press, Cambridge, MA, 1987.

[3] S. Geman and D. Geman. Stochastic relaxation, gibbs distribution and the bayesian restoration of images. *IEEE Trans. Pattern Anal. Mach. Intell.*, 6:721–741, 1984.

[4] P. E. Gill, W. Murray, and M. H. Wright. *Practical Optimization*. Academic Press, 1981.

[5] J. G. Harris, C. Koch, and J. Luo. A two-dimensional analog VLSI circuit for detecting discontinuities in early vision. *Science*, 248:1209–1211, 1990.

[6] J. G. Harris, C. Koch, J. Luo, and J. Wyatt. Resistive fuses: analog hardware for detecting discontinuities in early vision. In M. Mead, C.and Ismail, editor, *Analog VLSI Implementations of Neural Systems*. Kluwer, Norwell, MA, 1989.

[7] J.G. Harris. *Analog models for early vision*. PhD thesis, California Institute of Technology, Pasadena, CA, 1991. Dept. of Computation and Neural Systems.

[8] J.G. Harris, S.C. Liu, and B. Mathur. Discarding outliers in a nonlinear resistive network. In *International Joint Conference on Neural Networks*, pages 501–506, Seattle, WA., July 1991.

[9] P.J. Huber. *Robust Statistics*. J. Wiley & Sons, 1981.

[10] C. Koch, J. Marroquin, and A. Yuille. Analog "neuronal" networks in early vision. *Proc Natl. Acad. Sci. B. USA*, 83:4263–4267, 1987.

[11] J. Marroquin, S. Mitter, and T. Poggio. Probabilistic solution of ill-posed problems in computational vision. *J. Am. Statistic Assoc.*, 82:76–89, 1987.

[12] C. Mead. *Analog VLSI and Neural Systems*. Addison-Wesley, 1989.

[13] W. Millar. Some general theorems for non-linear systems possessing resistance. *Phil. Mag.*, 42:1150–1160, 1951.

[14] J. Platt. Constraint methods for neural networks and computer graphics. Dept. of Computer Science Technical Report Caltech-CS-TR-89-07, California Institute of Technology, Pasadena, CA, 1990.

[15] T. Poggio and C. Koch. An analog model of computation for the ill-posed problems of early vision. Technical report, MIT Artificial Intelligence Laboratory, Cambridge, MA, 1984. AI Memo No. 783.

[16] T. Poggio and C. Koch. Ill-posed problems in early vision: from computational theory to analogue networks. *Proc. R. Soc. Lond. B*, 226:303–323, 1985.

[17] B.G. Schunck. Robust computational vision. In *Robust methods in computer vision workshop.*, 1989.

[18] M. A. Sivilotti, M. A. Mahowald, and C. A. Mead. Real-time visual computation using analog CMOS processing arrays. In *1987 Stanford Conference on Very Large Scale Integration*, Cambridge, MA, 1987. MIT Press.
